# Correlated Bigram LSA for Unsupervised Language Model Adaptation

**Yik-Cheung Tam**[*]
InterACT, Language Technologies Institute
Carnegie Mellon University
Pittsburgh, PA 15213
yct@cs.cmu.edu

**Tanja Schultz**
InterACT, Language Technologies Institute
Carnegie Mellon University
Pittsburgh, PA 15213
tanja@cs.cmu.edu

## Abstract

We present a correlated bigram LSA approach for unsupervised LM adaptation for automatic speech recognition. The model is trained using efficient variational EM and smoothed using the proposed fractional Kneser-Ney smoothing which handles fractional counts. We address the scalability issue to large training corpora via bootstrapping of bigram LSA from unigram LSA. For LM adaptation, unigram and bigram LSA are integrated into the background N-gram LM via marginal adaptation and linear interpolation respectively. Experimental results on the Mandarin RT04 test set show that applying unigram and bigram LSA together yields 6%–8% relative perplexity reduction and 2.5% relative character error rate reduction which is statistically significant compared to applying only unigram LSA. On the large-scale evaluation on Arabic, 3% relative word error rate reduction is achieved which is also statistically significant.

## 1 Introduction

Language model (LM) adaptation is crucial to automatic speech recognition (ASR) as it enables higher-level contextual information to be effectively incorporated into a background LM improving recognition performance. Exploiting topical context for LM adaptation has shown to be effective for ASR using latent semantic analysis (LSA) such as LSA using singular value decomposition [1], Latent Dirichlet Allocation (LDA) [2, 3, 4] and HMM-LDA [5, 6]. One issue in LSA is the bag-of-word assumption which ignores word ordering. For document classification, word ordering may not be important. But in the LM perspective, word ordering is crucial since a trigram LM normally performs significantly better than a unigram LM for word prediction. In this paper, we investigate whether relaxing the bag-of-word assumption in LSA helps improving the ASR performance via LM adaptation.

We employ bigram LSA [7] which is a natural extension of LDA to relax the bag-of-word assumption by connecting the adjacent words in a document together to form a Markov chain. There are two main challenges in bigram LSA which are not addressed properly in [7] especially for large-scale application. Firstly, the model can be very sparse since it covers topical bigrams in $O(V^2 \cdot K)$ where $V$ and $K$ denote the vocabulary size and the number of topics. Therefore, model smoothing becomes critical. Secondly, model initialization is important for EM training, especially for bigram LSA due to the model sparsity. To tackle the first challenge, we represent bigram LSA as a set of $K$ topic-dependent backoff LM. We propose fractional Kneser-Ney smoothing [1] which supports

---

[*]This work is partly supported by the Defense Advanced Research Projects Agency (DARPA) under Contract No. HR0011-06-2-0001. Any opinions, findings and conclusions or recommendations expressed in this material are those of the authors and do not necessarily reflect the views of DARPA.

[1]This method was briefly mentioned in [8] without detail. To the best of our knowledge, our formulation in this paper is considered new to the research community.

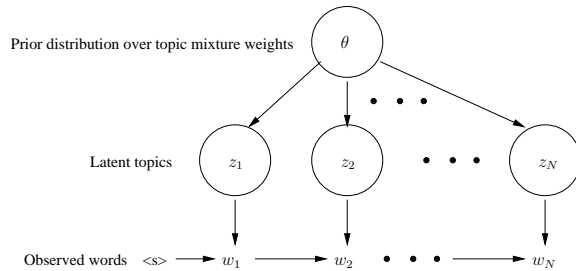

Figure 1: Graphical representation of bigram LSA. Adjacent words in a document are linked together to form a Markov chain from left to right.

fractional counts to smooth each backoff LM. We show that our formulation recovers the original Kneser-Ney smoothing [9] which supports only integral counts. To address the second challenge, we propose a bootstrapping approach for bigram LSA training using a well-trained unigram LSA as an initial model.

During unsupervised LM adaptation, word hypotheses from the first-pass decoding are used to estimate the topic mixture weight of each test audio to adapt both unigram and bigram LSA. The adapted unigram and bigram LSA are combined with the background LM in two stages. Firstly, marginal adaptation [10] is applied to integrate unigram LSA into the background LM. Then the intermediately adapted LM from the first stage is combined with bigram LSA via linear interpolation with the interpolation weights estimated by minimizing the word perplexity on the word hypotheses. The final adapted LM is employed for re-decoding.

Related work includes topic mixtures [11] which perform document clustering and train a trigram LM for each document cluster as an initial model. Sentence-level topic mixtures are modeled so that the topic label is fixed within a sentence. Topical N-gram model [12] focuses on phrase discovery and information retrieval. We do not apply this model because the phrase-based LM seems not outperform the word-based LM.

The paper is organized as follows: In Section 2, we describe the bigram LSA training and the fractional Kneser-Ney smoothing algorithm. In Section 3, we present the LM adaptation approach based on marginal adaptation and linear interpolation. In Section 4, we report LM adaptation results on Mandarin and Arabic ASR, followed by conclusions and future work in Section 5.

## 2   Correlated bigram LSA

Latent semantic analysis such as LDA makes a bag-of-word assumption that each word in a document is generated irrespective of its position in a document. To relax this assumption, bigram LSA has been proposed [7] to modify the graphical structure of LDA by connecting adjacent words in a document together to form a Markov chain. Figure 1 shows the graphical representation of bigram LSA where the top node represents the prior distribution over the topic mixture weights and the middle layer represents the latent topic label associated to each observed word at the bottom layer. The document generation procedure of bigram LSA is similar to LDA except that the previous word is taken into consideration for generating the current word:

1. Sample $\theta$ from a prior distribution $p(\theta)$
2. For each word $w_i$ at the $i$-th position of a document:
   (a) Sample topic label: $z_i \sim \text{Multinomial}(\theta)$
   (b) Sample $w_i$ given the previous word $w_{i-1}$ and the topic label $z_i$: $w_i \sim p(\cdot|w_{i-1}, z_i)$

Our incremental contributions for bigram LSA are three-folded: Firstly, we present a technique for topic correlation modeling using Dirichlet-Tree prior in Section 2.1. Secondly, we propose efficient algorithm for bigram LSA training via variational Bayes approach and model bootstrapping which are scalable to large settings in Section 2.2. Thirdly, we formulate the fractional Kneser-Ney smoothing to generalize the original Kneser-Ney smoothing which supports only integral counts in Section 2.3.

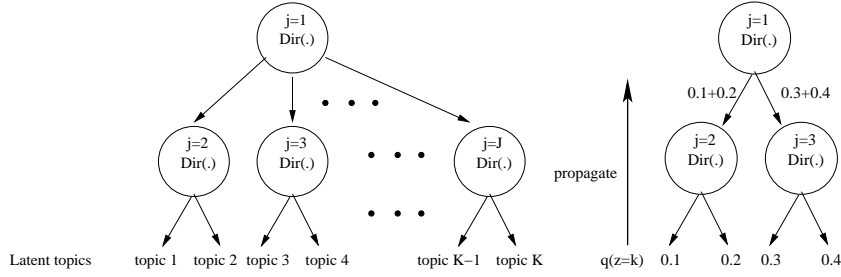

Figure 2: Left: Dirichlet-Tree prior of depth two. Right: Variational E-step as bottom-up propagation and summation of fractional topic counts.

## 2.1 Topic correlation

Modeling topic correlations is motivated by an observation that documents such as newspaper articles are usually organized into main-topic and sub-topic hierarchy for document browsing. From this perspective, a Dirichlet prior is not appropriate since it assumes topic independence. A Dirichlet-Tree prior [13, 14] is employed to capture topic correlations. Figure 2 (Left) illustrates a depth-two Dirichlet-Tree. A depth-one Dirichlet-tree is equivalent to a Dirichlet prior in LDA. The sampling procedure for the topic mixture weight $\theta \sim p(\theta)$ can be described as follows:

1. Sample a vector of branch probabilities $b_j \sim$ Dirichlet$(\cdot; \{\alpha_{jc}\})$ for each node $j = 1...J$ where $\{\alpha_{jc}\}$ denotes the parameter of the Dirichlet distribution at node $j$, i.e. the pseudo-counts of the outgoing branch $c$ at node $j$.

2. Compute the topic mixture weight as $\theta_k = \prod_{jc} b_{jc}^{\delta_{jc}(k)}$ where $\delta_{jc}(k)$ is an indicator function which sets to unity when the $c$-th branch of the $j$-th node leads to the leaf node of topic $k$ and zero otherwise. The $k$-th topic weight $\theta_k$ is computed as the product of sampled branch probabilities from the root node to the leaf node corresponding to topic $k$.

The structure and the number of outgoing branches of each Dirichlet node can be arbitrary. In this paper, we employ a balanced binary Dirichlet-tree.

## 2.2 Model training

Gibbs sampling was employed for bigram LSA training [7]. Despite the simplicity, it can be slow and inefficient since it usually requires many sampling iterations for convergence. We present a variational Bayes approach for model training. The joint likelihood of a document $w_1^N$, the latent topic sequence $z_1^N$ and $\theta$ using the bigram LSA can be written as follows:

$$p(w_1^N, z_1^N, \theta) = p(\theta) \cdot \prod_{i=1}^N p(z_i|\theta) \cdot p(w_i|w_{i-1}, z_i) \qquad (1)$$

By introducing a factorizable variational posterior distribution $q(z_1^N, \theta; \Gamma) = q(\theta) \cdot \prod_{i=1}^N q(z_i)$ over the latent variables and applying the Jensen's inequality, the lower bound of the marginalized document likelihood can be derived as follows:

$$\log p(w_1^N; \Lambda, \Gamma) = \log \int_\theta \sum_{z_1...z_N} q(z_1^N, \theta; \Gamma) \cdot \frac{p(w_1^N, z_1^N, \theta; \Lambda)}{q(z_1^N, \theta; \Gamma)} \qquad (2)$$

$$\geq \int_\theta \sum_{z_1...z_N} q(z_1^N, \theta; \Gamma) \cdot \log \frac{p(w_1^N, z_1^N, \theta; \Lambda)}{q(z_1^N, \theta; \Gamma)} \text{ (By Jensen's Inequality) } (3)$$

$$= E_q[\log \frac{p(\theta)}{q(\theta)}] + \sum_{i=1}^N E_q[\log \frac{p(z_i|\theta)}{q(z_i)}] + \sum_{i=1}^N E_q[\log p(w_i|w_{i-1}, z_i)] \qquad (4)$$

$$= Q(w_1^N; \Lambda, \Gamma) \qquad (5)$$

where the expectation is taken using the variational posterior $q(z_1^N, \theta)$. For the E-step, we compute the partial derivative of the auxiliary function $Q(\cdot)$ with respect to $q(z_i)$ and the parameter $\gamma_{jc}$ in the Dirichlet-Tree posterior $q(\theta)$. Setting the derivatives to zero yields:

**E-Steps:**

$$q(z_i = k) \quad \propto \quad p(w_i|w_{i-1}, k) \cdot e^{E_q[\log \theta_k; \{\gamma_{jc}\}]} \text{ for k = 1..K} \tag{6}$$

$$\gamma_{jc} \quad = \quad \alpha_{jc} + \sum_{i=1}^{N} E_q[\delta_{jc}(z_i)] = \alpha_{jc} + \sum_{i=1}^{N} \sum_{k=1}^{K} q(z_i = k) \cdot \delta_{jc}(k) \tag{7}$$

$$\text{where } E_q[\log \theta_k] \quad = \quad \sum_{jc} \delta_{jc}(k) \cdot E_q[\log b_{jc}] = \sum_{jc} \delta_{jc}(k) \left( \Psi(\gamma_{jc}) - \Psi(\sum_{c} \gamma_{jc}) \right) \tag{8}$$

where Eqn 7 is motivated from the conjugate property that the Dirichlet-Tree posterior given the topic sequence $z_1^N$ has the same form as the Dirichlet-Tree prior:

$$p(b_1^J|z_1^N) \quad \propto \quad p(z_1^N|b_1^J) \cdot p(b_1^J; \{\alpha_{jc}\}) \propto \left( \prod_{i=1}^{N} \prod_{jc} b_{jc}^{\delta_{jc}(z_i)} \right) \cdot \prod_{jc} b_{jc}^{\alpha_{jc}-1} \tag{9}$$

$$= \quad \prod_{jc} b_{jc}^{(\alpha_{jc}+\sum_{i=1}^{N} \delta_{jc}(z_i))-1} = \prod_{jc} b_{jc}^{\gamma'_{jc}-1} = \prod_{j=1}^{J} \text{Dirichlet}(b_j; \{\gamma'_{jc}\}) \tag{10}$$

Figure 2 (Right) illustrates that Eqn 7 can be implemented as propagation of fractional topic counts in a bottom-up fashion with each branch as an accumulator for $\gamma_{jc}$. Eqn 6 and Eqn 7 are applied iteratively until convergence is reached. For the M-step, we compute the partial derivative of the auxiliary function $Q(\cdot)$ over all training documents $d$ with respect to topic bigram probability $p(v|u,k)$ and set it to zero:

**M-Step (unsmoothed):**

$$p(v|u,k) \quad \propto \quad \sum_{d} \sum_{i=1}^{N_d} q(z_i = k|d) \cdot \delta(w_{i-1}, u)\delta(w_i, v) \tag{11}$$

$$= \quad \frac{\sum_{d} C_d(u,v|k)}{\sum_{d} \sum_{v'=1}^{V} C_d(u,v'|k)} = \frac{C(u,v|k)}{\sum_{v'=1}^{V} C(u,v'|k)} \tag{12}$$

where $N_d$ denote the number of words in document $d$ and $\delta(w_i, v)$ is a 0-1 Kronecker Delta function to test if the $i$-th word in document $d$ is vocabulary $v$. $C_d(u,v|k)$ denotes the fractional counts of a bigram $(u,v)$ belonging to topic $k$ in document $d$. Intuitively, Eqn 12 simply computes the relative frequency of the bigram $(u,v)$. However, this solution is not practical since bigram LSA assigns zero probability to unseen bigrams. Therefore, bigram LSA should be smoothed properly. One simple approach is to use Laplace-smoothing by adding a small count $\delta$ to all bigrams. However, this approach can lead to worse performance since it will bias the bigram probability towards a uniform distribution when the vocabulary size $V$ gets large. Our approach is to represent $p(v|u,k)$ as a standard backoff LM smoothed by fractional Kneser-Ney smoothing as described in Section 2.3.

Model initialization is crucial for variational EM training. We employ a bootstrapping approach using a well-trained unigram LSA as an initial model for bigram LSA so that $p(w_i|w_{i-1}, k)$ is approximated by $p(w_i|k)$ in Eqn 6. It saves computation and avoids keeping the full initial bigram LSA in memory during the EM training. To make the training procedure more practical, we apply bigram pruning during statistics accumulation in the M-step when the bigram count in a document is less than 0.1. This heuristic is reasonable since only a small number of topics are "active" to a bigram. With the sparsity, there is no need to store $K$ copies of accumulators for each bigram and thus reducing the memory requirement significantly. The pruned bigram counts are re-assigned to the most likely topic of the current document so that the counts are conserved. For practical implementation, accumulators are saved into the disk in batches for count merging. In the final step, each topic-dependent LM is smoothed individually using the merged count file.

## 2.3 Fractional Kneser-Ney smoothing

Standard backoff N-gram LM is widely used in the ASR community. The state-of-the-art smoothing for the backoff LM is based on Kneser-Ney smoothing [9]. The belief of its success is due to the preservation of marginal distributions. However, the original formulation only works for integral

counts which is not suitable for bigram LSA using fractional counts. Therefore, we propose the fractional Kneser-Ney smoothing as a generalization of the original formulation. The interpolated form using absolute discounting can be expressed as follows:

$$p_{KN}(v|u) \quad = \quad \frac{max\{C(u,v) - D, 0\}}{C(u)} + \lambda(u) \cdot p_{KN}(v) \tag{13}$$

where $D$ is a discounting factor. In the original formulation, $D$ lies between 0 and 1. But in our formulation, $D$ can be any positive number. Intuitively, $D$ controls the degree of smoothing. If $D$ is set to zero, the model is unsmoothed; If $D$ is too big, bigrams with counts smaller than $D$ are pruned from the LM. $\lambda(u)$ ensures the bigram probability sums to unity. After summing over all possible $v$ on both sides of Eqn 13 and re-arranging terms, $\lambda(u)$ becomes:

$$1 \quad = \quad \sum_v \frac{max\{C(u,v) - D, 0\}}{C(u)} + \lambda(u) \tag{14}$$

$$\implies \lambda(u) \quad = \quad 1 - \sum_v \frac{max\{C(u,v) - D, 0\}}{C(u)} = 1 - \sum_{v:C(u,v)>D} \frac{C(u,v) - D}{C(u)} \tag{15}$$

$$= \quad \frac{C(u) - \sum_{v:C(u,v)>D} C(u,v) + D \sum_{v:C(u,v)>D} 1}{C(u)} \tag{16}$$

$$= \quad \frac{\sum_{v:C(u,v) \leq D} C(u,v) + D \sum_{v:C(u,v)>D} 1}{C(u)} \tag{17}$$

$$= \quad \frac{C_{\leq D}(u, \cdot) + D \cdot N_{>D}(u, \cdot)}{C(u)} \tag{18}$$

where $C_{\leq D}(u, \cdot)$ denotes the sum of bigram counts following $u$ and smaller than $D$. $N_{>D}(u, \cdot)$ denotes the number of word types following $u$ with the bigram counts bigger than $D$.

In Kneser-Ney smoothing, the lower-order distribution $p_{KN}(v)$ is treated as unknown parameters which can be estimated using the preservation of marginal distributions:

$$\hat{p}(v) \quad = \quad \sum_u p_{KN}(v|u) \cdot \hat{p}(u) \tag{19}$$

where $\hat{p}(v)$ is the marginal distribution estimated from the background training data so that $\hat{p}(v) = \frac{C(v)}{\sum_{v'} C(v')}$. Therefore, we substitute Eqn 13 into Eqn 19:

$$C(v) \quad = \quad \sum_u \left( \frac{max\{C(u,v) - D, 0\}}{C(u)} + \lambda(u) \cdot p_{KN}(v) \right) \cdot C(u) \tag{20}$$

$$= \quad \left( \sum_u max\{C(u,v) - D, 0\} \right) + p_{KN}(v) \cdot \sum_u C(u) \cdot \lambda(u) \tag{21}$$

$$\implies p_{KN}(v) \quad = \quad \frac{C(v) - \sum_u max\{C(u,v) - D, 0\}}{\sum_u C(u) \cdot \lambda(u)} \tag{22}$$

$$= \quad \frac{C(v) - C_{>D}(\cdot, v) + D \cdot N_{>D}(\cdot, v)}{\sum_u C(u) \cdot \lambda(u)} \tag{23}$$

$$= \quad \frac{C_{\leq D}(\cdot, v) + D \cdot N_{>D}(\cdot, v)}{\sum_u C_{\leq D}(u, \cdot) + D \cdot N_{>D}(u, \cdot)} \text{ (using Eqn 18)} \tag{24}$$

$$= \quad \frac{C_{\leq D}(\cdot, v) + D \cdot N_{>D}(\cdot, v)}{\sum_v C_{\leq D}(\cdot, v) + D \cdot N_{>D}(\cdot, v)} \tag{25}$$

Eqn 25 generalizes Kneser-Ney smoothing to integral and fractional counts. For the original formulation, $C_{\leq D}(u, \cdot)$ equals to zero since each observed bigram count must be at least one by definition with $D$ less than one. As a result, the $D$ term cancels out yielding the original formulation which counts the number of words preceding $v$ and thus recovering the original formulation. Intuitively, the numerator in Eqn 25 measures the total discounts of observed bigrams ending at $v$. In other words, fractional Kneser-Ney smoothing estimates the lower-order probability distribution using the relative frequency over *discounts* instead of word counts. With this approach, each topic-dependent LM in bigram LSA can be smoothed using our formulation.

# 3 Unsupervised LM adaptation

Unsupervised LM adaptation is performed by first inferring the topic distribution of each test audio using the word hypotheses from the first-pass decoding via variational inference in Eqn 6–7. Relative frequency over the branch posterior counts $\gamma_{jc}$ is applied on each Dirichlet node $j$. The MAP topic mixture weight $\hat{\theta}$ and the adapted unigram and bigram LSA are computed as follows:

$$\hat{\theta}_k \quad \propto \quad \prod_{jc} \left( \frac{\gamma_{jc}}{\sum_{c'} \gamma_{jc'}} \right)^{\delta_{jc}(k)} \quad \text{for } k = 1...K \tag{26}$$

$$p_a(v) \quad = \quad \sum_{k=1}^{K} p(v|k) \cdot \hat{\theta}_k \text{ and } p_a(v|u) = \sum_{k=1}^{K} p(v|u,k) \cdot \hat{\theta}_k \tag{27}$$

The unigram LSA marginals are integrated into the background N-gram LM $p_{bg}(v|h)$ via marginal adaptation [10] as follows:

$$p_a^{(1)}(v|h) \quad \propto \quad \left( \frac{p_a(v)}{p_{bg}(v)} \right)^{\beta} \cdot p_{bg}(v|h) \tag{28}$$

Marginal adaptation has a close connection to maximum entropy modeling since the marginal constraints can be encoded as unigram features. Intuitively, bigram LSA would be integrated in the same fashion by introducing bigram marginal constraints. However, we found that integrating bigram features via marginal adaptation did not offer further improvement compared to only integrating unigram features. Since marginal adaptation integrates a unigram feature as a likelihood ratio between the adapted marginal $p_a(v)$ and the background marginal $p_{bg}(v)$ in Eqn 28, perhaps the unigram and bigram likelihood ratios are very similar and thus the latter does not give extra information. Another explanation is that marginal adaptation corresponds to only one iteration of generalized iterative scaling (GIS). Due to the large number of bigram features in terms of millions, one GIS iteration may not be sufficient for convergence. On the other hand, simple linear LM interpolation is found to be effective in our experiment. The final LM adaptation formula is provided using results from Eqn 27 and Eqn 28 as a two-stage process:

$$p_a^{(2)}(v|h) \quad = \quad \lambda \cdot p_a^{(1)}(v|h) + (1 - \lambda) \cdot p_a(v|u) \tag{29}$$

where $\lambda$ is tuned to optimize perplexity on word hypotheses from the first-pass decoding on a per-audio basis.

# 4 Experimental setup

Our LM adaptation approach was evaluated using the RT04 Mandarin Broadcast News evaluation system. The system employed context-dependent Initial-Final acoustic models trained using 100-hour broadcast news audio from the Mandarin HUB4 1997 training set and a subset of TDT4. 42-dimension features were extracted after linear discriminant analysis projected from a window of MFCC and energy features. The system employed a two-pass decoding strategy using speaker-independent and speaker-adaptive acoustic models. For the second-pass decoding, we applied standard acoustic model adaptation such as vocal tract length normalization and maximum likelihood linear regression on the feature and model spaces. The training corpora include Xinhua News 2002 (January–September) containing 13M words and 64k documents. A background 4-gram LM was trained using modified Kneser-Ney smoothing using the SRILM toolkit [15]. The same training corpora were used for unigram and bigram LSA training with 200 topics. The vocabulary size is 108k words. Discounting factor $D$ for fractional Kneser-Ney smoothing was set to $0.4$.

First-pass decoding was first performed to obtain an automatic transcript for each audio show. Then unsupervised LM adaptation was applied using the automatic transcript to obtain an adapted LM for second-pass decoding using the approach described in Section 3. Word perplexity and character error rates (CER) were measured on the Mandarin RT04 test set. Matched pairs sentence-segment word error test was performed for significance test using the NIST scoring tool.

Table 1: Correlated bigram topics extracted from bigram LSA.

| Topic index | Top bigrams sorted by $p(u,v|k)$ |
|---|---|
| "topic-61" | 的+学生('s student), 的+教育('s education), 教育+的(education 's) 学校+的(school 's), 少年+班(youth class), 素质+教育(quality of education) |
| "topic-62" | 人才+培养(expert cultivation), 大学+校长(university chancellor) 着+名(famous), 所+高校(high-school), 的+学生('s student) |
| "topic-63" | 和+社会保障(and social security), 的+就业('s employment), 失业+人员(unemployed officer), 就业+岗位(employment position) |
| "topic-64" | 的+研究('s research), 专家+学者(expert people), 等+领域(etc area) 生物+技术(biological technology), 研究+成果(research result) |
| "topic-65" | 人类+基因组(Human DNA sequence), 的+基因('s DNA) 生物+技术(biological technology), 胚胎+干细胞(embryo stem cell) |

Table 2: Character Error Rates (Word perplexity) on the RT04 test set. Bigram LSA was applied in addition to unigram LSA.

| LM (13M) | CCTV | NTDTV | RFA | OVERALL |
|---|---|---|---|---|
| background LM | 15.3% (748) | 21.8 (1718) | 39.5 (3655) | 24.9 |
| +unigram LSA | 14.4 (629) | 21.5 (1547) | 38.9 (3015) | 24.3 |
| +bigram LSA (Kneser-Ney, 30 topics) | 14.5 (604) | **20.7** (1502) | 39.0 (2736) | 24.1 |
| +bigram LSA (Witten-Bell) | 14.1 (594) | 20.9 (1452) | 38.3 (2628) | 23.8 |
| +bigram LSA (Kneser-Ney) | **14.0** (587) | 20.8 (1448) | **38.2** (2586) | **23.7** |

## 4.1 LM adaptation results

Table 1 shows the correlated bigram topics sorted by the joint bigram probability $p(v|u,k) \cdot p(u|k)$. Most of the top bigrams appear either as phrases or words attached with a stopword such as 的('s in English). Table 2 shows the LM adaptation results in CER and perplexity. Applying both unigram and bigram LSA yields consistent improvement over unigram LSA in the range of 6.4%–8.5% relative reduction in perplexity and 2.5% relative reduction in the overall CER. The CER reduction is statistically significant at 0.1% significance level. We compared our proposed fractional Kneser-Ney smoothing with Witten-Bell smoothing which also supports fractional counts. The results showed that Kneser-Ney smoothing performs slightly better than Witten-Bell smoothing. Increasing the number of topics in bigram LSA helps despite model sparsity. We applied extra EM iterations on top of the bootstrapped bigram LSA but no further performance improvement was observed.

## 4.2 Large-scale evaluation

We evaluated our approach using the CMU-InterACT vowelized Arabic transcription system discriminatively trained on 1500-hour transcribed audio using MMIE for the GALE Phase-3 evaluation. A large background 4-gram LM was trained using 962M-word text corpora with 737k vocabulary. Unigram and bigram LSA were trained on the same corpora and were applied to lattice rescoring on Dev07 and unseen Dev08 test sets with 2.6-hour and 3-hour audio shows containing broadcast news (BN) and broadcast conversation (BC) genre. Table 3 shows that bigram LSA rescoring reduces the overall word error rate by more than 3.0% relative compared to the unadapted baseline on both sets which are statistically significant at 0.1% significance level. However, degradation is observed using trigram LSA compared to bigram LSA which may be due to data sparseness.

Table 3: Lattice rescoring results in word error rate on Dev07 (unseen Dev08) using the CMU-InterACT Arabic transcription system for the GALE Phase-3 evaluation.

| GALE LM (962M) | BN | BC | OVERALL |
|---|---|---|---|
| background LM | 11.6% | 19.4 | 14.3 (16.4) |
| +unigram LSA | 11.5 | 19.2 | 14.2 (16.3) |
| +bigram LSA (Witten-Bell) | **11.0** | 19.0 | 13.9 (15.9) |
| +bigram LSA (Kneser-Ney) | **11.0** | 18.9 | **13.8 (15.9)** |
| +trigram LSA (Kneser-Ney) | 11.3 | **18.8** | 14.0 (-) |

# 5 Conclusion

We present a correlated bigram LSA approach for unsupervised LM adaptation for ASR. Our contributions include efficient variational EM for model training and fractional Kneser-Ney approach for LM smoothing with fractional counts. Bigram LSA yields additional improvement in both perplexity and recognition performance in addition to unigram LSA. Increasing the number of topics for bigram LSA helps despite the model sparsity. Bootstrapping bigram LSA from unigram LSA saves computation and memory requirement during EM training. Our approach is scalable to large training corpora and works well on different languages. The improvement from bigram LSA is statistically significant compared to the unadapted baseline. Future work includes applying the proposed approach for statistical machine translation.

## Acknowledgement

We would like to thank Mark Fuhs for help parallelizing the bigram LSA training via condor.

## References

[1] J. R. Bellegarda, "Large Vocabulary Speech Recognition with Multispan Statistical Language Models," *IEEE Transactions on Speech and Audio Processing*, vol. 8, no. 1, pp. 76–84, Jan 2000.

[2] D. Blei, A. Ng, and M. Jordan, "Latent Dirichlet Allocation," in *Journal of Machine Learning Research*, 2003, pp. 1107–1135.

[3] Y. C. Tam and T. Schultz, "Language model adaptation using variational Bayes inference," in *Proceedings of Interspeech*, 2005.

[4] D. Mrva and P. C. Woodland, "Unsupervised language model adaptation for mandarin broadcast conversation transcription," in *Proceedings of Interspeech*, 2006.

[5] T. Griffiths, M. Steyvers, D. Blei, and J. Tenenbaum, "Integrating topics and syntax," in *Advances in Neural Information Processing Systems*, 2004.

[6] B. J. Hsu and J. Glass, "Style and topic language model adaptation using HMM-LDA," in *Proceedings of Empirical Methods on Natural Language Processing (EMNLP)*, 2006.

[7] Hanna M. Wallach, "Topic Modeling: Beyond Bag-of-Words," in *International Conference on Machine Learning*, 2006.

[8] P. Xu, A. Emami, and F. Jelinek, "Training connectionist models for the structured language model," in *Proceedings of Empirical Methods on Natural Language Processing (EMNLP)*, 2003.

[9] R. Kneser and H. Ney, "Improved backing-off for M-gram language modeling," in *Proceedings of the IEEE International Conference on Acoustics, Speech, and Signal Processing (ICASSP)*, 1995, vol. 1, pp. 181–184.

[10] R. Kneser, J. Peters, and D. Klakow, "Language model adaptation using dynamic marginals," in *Proceedings of European Conference on Speech Communication and Technology (EUROSPEECH)*, 1997, pp. 1971–1974.

[11] R. Iyer and M. Ostendorf, "Modeling long distance dependence in language: Topic mixtures versus dynamic cache models," *IEEE Transactions on Speech and Audio Processing*, vol. 7, no. 1, pp. 30–39, Jan 1999.

[12] X. Wang, A. McCallum, and X. Wei, "Topical N-grams: Phrase and topic discovery, with an application to information retrieval," in *IEEE International Conference on Data Mining*, 2007.

[13] T. Minka, "The dirichlet-tree distribution," 1999.

[14] Y. C. Tam and T. Schultz, "Correlated latent semantic model for unsupervised language model adaptation," in *Proceedings of the IEEE International Conference on Acoustics, Speech, and Signal Processing (ICASSP)*, 2007.

[15] A. Stolcke, "SRILM - an extensible language modeling toolkit," in *Proceedings of International Conference on Spoken Language Processing (ICSLP)*, 2002.
